# An Attractor Neural Network Model of Recall and Recognition

**Eytan Ruppin**
Department of Computer Science
School of Mathematical Sciences
Sackler Faculty of Exact Sciences
Tel Aviv University
69978, Tel Aviv, Israel

**Yechezkel Yeshurun**
Department of Computer Science
School of Mathematical Sciences
Sackler Faculty of Exact Sciences
Tel Aviv University
69978, Tel Aviv, Israel

## Abstract

This work presents an Attractor Neural Network (ANN) model of Recall and Recognition. It is shown that an ANN model can qualitatively account for a wide range of experimental psychological data pertaining to the these two main aspects of memory access. Certain psychological phenomena are accounted for, including the effects of list-length, word-frequency, presentation time, context shift, and aging. Thereafter, the probabilities of successful Recall and Recognition are estimated, in order to possibly enable further quantitative examination of the model.

## 1 Motivation

The goal of this paper is to demonstrate that a Hopfield-based [Hop82] ANN model can qualitatively account for a wide range of experimental psychological data pertaining to the two main aspects of memory access, Recall and Recognition. *Recall* is defined as the ability to retrieve an item from a list of items (words) originally presented during a previous learning phase, given an appropriate cue (*cued Recall*), or spontaneously (*free Recall*). *Recognition* is defined as the ability to successfully acknowledge that a certain item has or has not appeared in the tutorial list learned before.

The main prospects of ANN modeling is that some parameter values, that in former, 'classical' models of memory retrieval (see e.g. [GS84]) had to be explicitly assigned, can now be shown to be emergent properties of the model.

## 2   The Model

The model consists of a Hopfield ANN, in which distributed patterns representing the learned items are stored during the learning phase, and are later presented as inputs during the test phase. In this framework, successful Recall and Recognition is defined. Some additional components are added to the basic Hopfield model to enable the modeling of the relevant psychological phenomena.

### 2.1   The Hopfield Model

The Hopfield model's dynamics are composed of a non-linear, iterative, asynchronous transformation of the network state [Hop82]. The process may include a stochastic noise which is analogous to the 'temperature' $T$ in statistical mechanics. Formally, the Hopfield model is described as follows: Let neuron's $i$ state be a binary variable $S_i$, taking the values $\pm 1$ denoting a firing or a resting state, correspondingly. Let the network's state be a vector $S$ specifying the binary values of all its neurons. Let $J_{ij}$ be the synaptic strength between neurons $i$ and $j$. Then, $h_i$, the input 'field' of neuron $i$ is given by $h_i = \sum_{j \neq i}^{N} J_{ij} S_j$. The neuron's dynamic behavior is described by

$$S_i(t+1) = \begin{cases} 1, & \text{with probability } \frac{1}{2}(1 + tgh(\frac{h_i}{T})) \\ -1, & \text{with probability } \frac{1}{2}(1 - tgh(\frac{h_i}{T})) \end{cases}$$

Storing a new memory pattern $\xi^\mu$ in the network is performed by modifying every $ij$ element of the synaptic connection matrix according to $J_{ij}^{new} = J_{ij}^{old} + \frac{1}{N}\xi^\mu{}_i \xi^\mu{}_j$.

A Hopfield network will always converge to a stable state, and every stored memory is an *attractor* having an area surrounding it termed its *basin of attraction* [Hop82]. In addition to the stored memories, also other, non-memory states exist as stable states (local minima) of the network [AGS85]. The maximal number $m$ of (randomly generated) memory patterns which can be stored in the basic Hopfield network of $n$ neurons is $m = \alpha_c \cdot n$, $\alpha_c \approx 0.14$ [AGS85].

### 2.2   Recall and Recognition in the model's framework

#### 2.2.1   Recall

*Recall* is considered successful when upon starting from an initial cue the network converges to a stable state which corresponds to the learned memory nearest to the input pattern. Inter-pattern distance is measured by the Hamming distance between the input and the learned item encodings. If the network converges to a non-memory stable state, its output will stand for a 'failure of recall' response. [1].

### 2.2.2 Recognition

*Recognition* is considered successful when the network arrives at a stable state during a time interval $\Delta$, beginning from input presentation. In general, the shorter the distance between an input and its nearest memory, the faster is its convergence [AM88, KP88, RY90]. Since non-memory (non-learned) stable states have higher energy levels and much shallower basins of attraction than memorized stable states [AGS85, LN89], convergence to such states takes significantly longer timer. Therefore, there exists a range of possible values of $\Delta$ that enable successful recognition only of inputs similar to one of the stored memories.

### 2.3  Other features of the model

- The context of the psychological experiments is represented as a substring of the input's encoding. In order to minimize inter-pattern correlation, the size of the context encoding relative to the total size of the memory encoding is kept small.

- The total associational linkage of a learned item, is modeled as an external field vector $E$. When a learned memory pattern $\xi^\mu$ is presented to the network, the value of the external field vector generated is $E_i = h \cdot \xi^\mu$, where $h$ is an 'orientation' coefficient, expressing the association strength.

Additional features, including a modified storage equation accounting for learning taking place at the test phase, and a storage decay parameter, are described in [RY90].

## 3   The Modeling of experimental data.

Regarding every phenomenon discussed, a brief description of the psychological findings is followed by an account of its modeling. We rely on the known results pertaining to Hopfield models to show that qualitatively, the psychological phenomena reviewed are emergent properties of the model. When such analytical evidence is lacking, simulations were performed in order to account for the experimental data. For a review of the psychological literature supporting the findings modeled see [GS84].

**The List-Length Effect:** It is known that the probability of successful Recall or Recognition of a particular item decreases as the length of list of learned items increases.

List length is expressed in memory load. Since It has been shown that the width of the memories basins of attraction monotonically decreases following an approximately inverse parabolic curve [Wei85], Recall performance should decrease as memory load is increased. We have examined the convergence time of the same set of input patterns at different values of memory load. As demonstrated in Fig. 1, it was found that, as the memory load is increased, successful convergence has occurred (on the average) only after an increasingly growing number of asynchronous iterations. Hence, convergence takes more time and can result in Recognition failure, although memories' stability is maintained till the critical capacity $\alpha_c$ is reached.

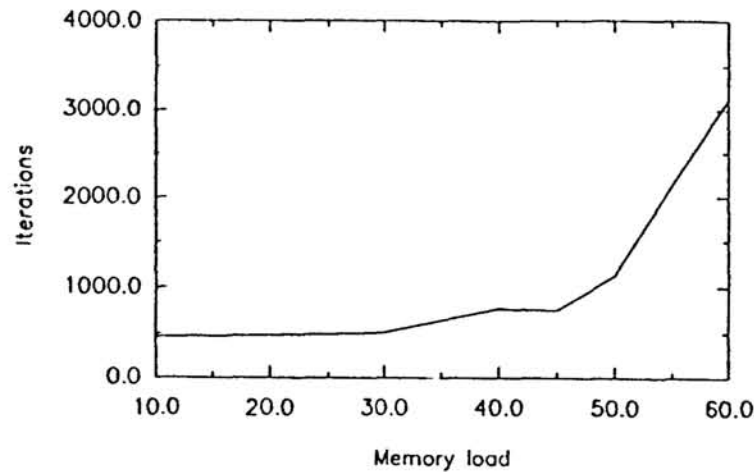

Figure 1: Recognition speed (No. of asynchronous iterations) as a function of memory load (No. of stored memories). The network's parameters are $n = 500, T = 0.28$

**The word-frequency effect:** The more frequent a word is in language, the probability of recalling it increases, while the probability of recognizing it decreases.

A word's frequency in the language is assumed to effect its retrieval through the stored word's semantic relations and associations [Kat85, NCBK87]. It is assumed, that relative to low frequency words, high frequency words have more semantic relations and therefore more connections between the patterns representing them and other patterns stored in the memory (i.e., in other networks). This one-to-many relationship is assumed to be reciprocal, i.e., each of the externally stored patterns has also connections projected to several of the stored patterns in the allocated network.

The process leading to the formation of the external field $E$ (acting upon the allocated network), generated by an input pattern nearest to some stored memory pattern $\xi^\mu$ is assumed to be characterized as follows:

1. There is a threshold degree of overlap $\theta_{min}$, such that $E > 0$ only when the allocated network's state overlap $H^\mu$ is higher than $\theta_{min}$.

2. At overlap values $H^\mu$ which are only moderately larger than $\theta_{min}$, $h^\mu$ is monotonically increasing, but as $H^\mu$ continues to rise, a certain 'optimal' point is reached, beyond which $h^\mu$ is monotonically decreasing.

3. High-frequency words have lower $\theta_{min}$ values than low-frequency words.

Recognition tests are characterized by a high initial value of overlap $H^\mu$, to some memory $\xi^\mu$. The value of $h^\mu$ and $E^\mu$ generated is post-optimal and therefore smaller than in the case of low-frequency words which have higher $\theta_{min}$ values.

In Recall tests the initial situation is characterized by low values of overlap $H^\mu$ to some nearest memory $\xi^\mu$. only the overlap value of high-frequency words is sufficient for activating associated items, i.e. $H^\mu > \theta_{min}$.

**Presentation Time:** Increasing the presentation time of learned words is known to improve both their Recall and Recognition.

This is explained by the phenomenon of maintenance rehearsal; The memories' basins of attraction get deeper, since the 'energy' $E$ of a given state equals to $\sum_{\mu=1}^{m} H^{\mu 2}$. Deeper basins of attraction are also wider [HFP83, KPKP90]. Therefore, the probability of successful Recall and Recognition of rehearsed items is increased. The effect of a uniform rehearsal is equivalent to a temperature decrease. Hence, increasing presentation time will attenuate and delay the List length phenomenon, till a certain limit. In a similar way, the Test Delay phenomenon is accounted for [RY90].

**Context Shift:** The term Context Shift refers to the change in context from the tutorial period to the test period. Studies examining the effect of context shift have shown a decrement in Recall performance with context shift, but little change in Recognition performance.

As demonstrated in [RY90], when a context shift is simulated by flipping some of the context string's bits, Recall performance severely deteriorates while memories stability remains intact. No significant increase in the time (i.e. number of asynchronous iterations) required for convergence was found, thus maintaining the pre-shift probability of successful Recognition.

**Age differences in Recall and Recognition:** It was found that older people perform more poorly on Recall tasks than they do on Recognition tasks [CM87]. These findings can be accounted for by the assumption that synapses are being weakened and deleted with aging, which although being controversial has gained some experimental support (see [RY90]). We have investigated the retrieval performance as a function of the input's initial overlap, various levels of synaptic dilution, and memory load: As demonstrated in Fig. 2, when the synaptic dilution is increased, a 'critical' phase is reached where memory retrieval of far-away input patterns is decreased but the retrieval of input patterns with a high level of initial overlap remains intact. As the memory load is increased, this 'critical' phase begins at lower levels of synaptic dilution. On the other hand, only a mild increase (of 15%) in recognition speed was found.

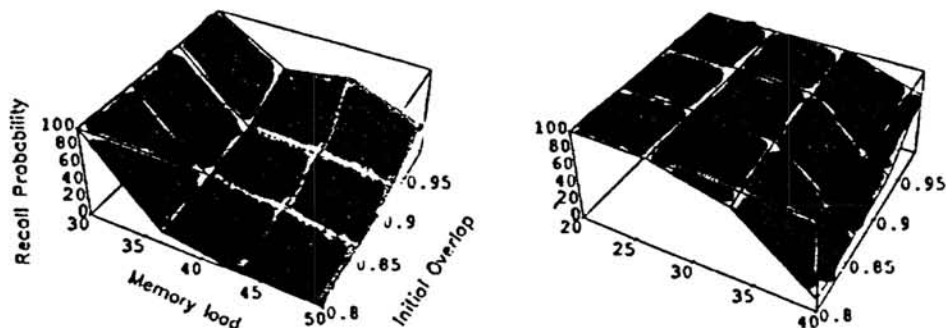

Figure 2: The probability of successful retrieval performance as a function of memory load and the input pattern's initial overlap, at two different degrees of synaptic dilution; 50% in the right-sided figure, and 55% in the left-sided figure. The network's parameters are $n = 500$, $T = 0.05$.

The interested reader can find a description of the modeling of additional phenomena, including test position, word fragment completion, and distractor similarity, in [RY90].

# 4  On a quantitative test of the model.

## 4.1  Estimating Recall performance

In a given network, with $n$ neurons and $m$ memories, the radius $r$ of the basins of attraction of the memories decreases as the memory load parameter ($\alpha = m/n$) is increased. According to [MPRV87], $n, m$, and $r$ are related according to the expression $m = \frac{(1-2 \cdot r)^2}{4} \cdot \frac{n}{\log n}$.

The concept of the basins of attraction implies a non-linear probability function with low probability when input vectors are further than the radius of attraction and high probability otherwise. The slope of this non-linearity increases as the noise level $T$ is decreased.

The probability $P_c$ that a *random* input vector will converge to one of the stored memories can be estimated by $P_c \approx \frac{\sum_{d=1}^{rn} \binom{n}{d}}{2^n} \cdot m$. It is interesting to note that the rates of change of $r$ and of $P_c$ have distinct forms; Recall tests beginning from randomly generated cues would yield a very low rate of successful Recall ($P_c$). Yet, if one examines Recall by picking a stored memory, flipping some of its encoding bits, and presenting it as an input to the network (determining $r$), 'reasonable' levels of successful Recall can still be obtained even when a 'considerable' number of encoding bits are flipped. $P_c$ can be also estimated by considering the context representation [RY90].

## 4.2  Estimating Recognition performance

The probability of correct Recognition depends mainly on the the length of the interval $\Delta$; assume that after an input pattern is presented to a network of $n$ neurons, during the time interval $\Delta$, $k$ iterations steps of a Monte Carlo simulation are performed: In each such step, a neuron is randomly selected, and then it examines whether or not it should flip its state, according to its input.

We show that the probability $P_g\{d\}$ that an input pattern will be successfully recognized is is bounded by $P_g\{d\} \geq 1 - d \cdot e^{\frac{-k}{n}}$. It can be seen that Recognition's success depends strongly on the initial input proximity to a stored memory, and even more strongly dependent on the number of allowed asynchronous iteration $k$, determined by the length of $\Delta$. For a selection of $k = n(ln(d) + c)$, one obtains $P_g \geq 1 - e^{-c}$. The expected number of iterations, (denoted as $Exp(X)$) till successful convergence is achieved is $E(X) = \sum_{i=1}^{d} E(X_i) = n \cdot \sum_{i=1}^{d} \frac{1}{i} \approx n \cdot ln(d)$.

In the more general case, Let $o$ denote the Hamming distance (between the network's state $S$ and a stored memory) below which retrieval is considered successful. Then, the corrected estimations of retrieval performance are $P_g \geq 1 - \binom{d}{o} \cdot e^{\frac{-k \cdot o}{n}}$, and $E(X) \approx n \cdot ln(\frac{d}{o})$. In simulations we have performed, ($n = 500, d = 20, o = 10$), the

average number of iterations until successful convergence was in the range of 300 -
400, in excellent correspondence with the predicted expectation, $E(X) = 500 \cdot ln(2)$.

## Footnotes

[1] The question of "How do such non-memory states bear the meaning of 'recall failure'?" is out of the scope of this work. However, a possible explanation is that during the learning phase 'meaning' is assigned to the stored patterns via connections formed with external patterns, and since non-memory states lack such associations with external patterns, they are 'meaningless', yielding the 'recall failure' response. Another possible mechanism is that every output pattern generated in the recall process passes also a recognition phase so that non-memory states are rejected, (see the following paragraph describing recognition in our model).

# References

[AGS85]    D. J. Amit, H. Gutfreund, and H. Sompolinsky. Storing infinite numbers
of patterns in a spin-glass model of neural networks. *Phys. Rev. Lett.*,
55:1530, 1985.

[AM88]     S. I. Amari and K. Maginu. Statistical neurodynamics of associative
memory. *Neural Networks*, 1:63, 1988.

[CM87]     F.I.M. Craik and J.M. McDowd. Age differences in recall and recog-
nition. *Journal of Experimental Psychology; Learning, Memory, and
Cognition*, 13(3):474, 1987.

[GS84]     G. Gillund and M. Shiffrin. A retrieval model for both recognition and
recall. *Psychological Review*, 91:1, 1984.

[HFP83]    J.J. Hopfield, D. I. Fienstien, and R. G. Palmer. Unlearning' has a
stabilizing effect in collective memories. *Nature*, 304:158, 1983.

[Hop82]    J.J. Hopfield. Neural networks and physical systems with emergent
collective abilities. *Proc. Nat. Acad. Sci. USA*, 79:2554, 1982.

[Kat85]    T. Kato. Semantic-memory sources of episodic retrieval failure. *Memory
& Cognition*, 13(5):442, 1985.

[KP88]     J. Komlós and R. Paturi. Convergence results in an associative memory
model. *Neural Networks*, 1:239, 1988.

[KPKP90]   B. Kagmar-Parsi and B. Kagmar-Parsi. On problem solving with hop-
field neural networks. *Biol. Cybern.*, 62:415, 1990.

[LN89]     M. Lewenstein and A. Nowak. Fully connected neural networks with
self-control of noise levels. *Phys. Rev. Lett.*, 62(2):225, 1989.

[MPRV87]   R.J. McEliece, E.C. Posner, E.R. Rodemich, and S.S. Venkatesh. The
capacity of the hopfield associative memory. *IEEE Transactions on
Information theory*, IT-33(4):461, 1987.

[NCBK87]   D.L Nelson, J.J. Canas, M.T. Bajo, and P.D. Keelan. Comparing word
fragment completion and cued recall with letter cues. *Journal of Exper-
imental Psychology: Learning, Memory and Cognition*, 13(4):542, 1987.

[RY90]     E. Ruppin and Y. Yeshurun. Recall and recognition in an attractor
neural network model of memory retrieval. Technical report, Dept. of
Computer Science, Tel-Aviv University, 1990.

[Wei85]    G. Weisbuch. Scaling laws for the attractors of hopfield networks. *J.
Physique Lett.*, 46:L–623, 1985.